# Model Selection in Gaussian Graphical Models: High-Dimensional Consistency of $\ell_1$-regularized MLE

**Pradeep Ravikumar**[†], **Garvesh Raskutti**[†], **Martin J. Wainwright**[†*] **and Bin Yu**[†*]
Department of Statistics[†], Department of EECS[*],
University of California, Berkeley
{pradeepr,garveshr,wainwright,binyu}@stat.berkeley.edu

## Abstract

We consider the problem of estimating the graph structure associated with a Gaussian Markov random field (GMRF) from i.i.d. samples. We study the performance of the $\ell_1$-regularized maximum likelihood estimator in the high-dimensional setting, where the number of nodes in the graph $p$, the number of edges in the graph $s$ and the maximum node degree $d$, are allowed to grow as a function of the number of samples $n$. Our main result provides sufficient conditions on $(n, p, d)$ for the $\ell_1$-regularized MLE estimator to recover all the edges of the graph with high probability. Under some conditions on the model covariance, we show that model selection can be achieved for sample sizes $n = \Omega(d^2 \log(p))$, with the error decaying as $\mathcal{O}(\exp(-c \log(p)))$ for some constant $c$. We illustrate our theoretical results via simulations and show good correspondences between the theoretical predictions and behavior in simulations.

## 1 Introduction

The area of high-dimensional statistics deals with estimation in the "large $p$, small $n$" setting, where $p$ and $n$ correspond, respectively, to the dimensionality of the data and the sample size. Such high-dimensional problems arise in a variety of applications, among them remote sensing, computational biology and natural language processing, where the model dimension may be comparable or substantially larger than the sample size. It is well-known that such high-dimensional scaling can lead to dramatic breakdowns in many classical procedures. In the absence of additional model assumptions, it is frequently impossible to obtain consistent procedures when $p \gg n$. Accordingly, an active line of statistical research is based on imposing various restrictions on the model—-for instance, sparsity, manifold structure, or graphical model structure—-and then studying the scaling behavior of different estimators as a function of sample size $n$, ambient dimension $p$ and additional parameters related to these structural assumptions.

In this paper, we study the problem of estimating the graph structure of a Gauss Markov random field (GMRF) in the high-dimensional setting. This graphical model selection problem can be reduced to the problem of estimating the zero-pattern of the inverse covariance or concentration matrix $\Theta^*$. A line of recent work [1, 2, 3, 4] has studied estimators based on minimizing Gaussian log-likelihood penalized by the $\ell_1$ norm of the entries (or the off-diagonal entries) of the concentration matrix. The resulting optimization problem is a log-determinant program, which can be solved in polynomial time with interior point methods [5], or by faster co-ordinate descent algorithms [3, 4]. In recent work, Rothman et al. [1] have analyzed some aspects of high-dimensional behavior, in particular establishing consistency in Frobenius norm under certain conditions on the model covariance and under certain scalings of the sparsity, sample size, and ambient model dimension.

The main contribution of this paper is to provide sufficient conditions for model selection consistency of $\ell_1$-regularized Gaussian maximum likelihood. It is worth noting that such a consistency result for structure learning of Gaussian graphical models cannot be derived from Frobenius norm consistency alone. For any concentration matrix $\Theta$, denote the set of its non-zero off-diagonal entries

by $E(\Theta) = \{s \neq t \mid \Theta_{st} \neq 0\}$. (As will be clarified below, the notation $E$ alludes to the fact that this set corresponds to the edges in the graph defining the GMRF.) Under certain technical conditions to be specified, we prove that the $\ell_1$-regularized (on off-diagonal entries of $\Theta$) Gaussian MLE recovers this edge set with high probability, meaning that $\mathbb{P}[E(\widehat{\Theta}) = E(\Theta^*)] \rightarrow 1$. In many applications of graphical models (e.g., protein networks, social network analysis), it is this edge structure itself, as opposed to the weights $\Theta^*_{st}$ on the edges, that is of primary interest. Moreover, we note that model selection consistency is useful even when one is interested in convergence in spectral or Frobenius norm; indeed, having extracted the set $E(\Theta^*)$, we could then restrict to this subset, and estimate the non-zero entries of $\Theta^*$ at the faster rates applicable to the reduced dimension.

The remainder of this paper is organized as follows. In Section 2, we state our main result, discuss its connections to related work, and some of its consequences. Section 3 provides an outline of the proof. In Section 4, we provide some simulations that illustrate our results.

**Notation** For the convenience of the reader, we summarize here notation to be used throughout the paper. Given a vector $u \in \mathbb{R}^d$ and parameter $a \in [1, \infty]$, we use $\|u\|_a$ to denote the usual $\ell_a$ norm. Given a matrix $U \in \mathbb{R}^{p \times p}$ and parameters $a, b \in [1, \infty]$, we use $\|U\|_{a,b}$ to denote the induced matrix-operator norm $\max_{\|y\|_a=1} \|Uy\|_b$; see [6] for background. Three cases of particular importance in this paper are the *spectral norm* $\|U\|_2$, corresponding to the maximal singular value of $U$; the $\ell_\infty/\ell_\infty$-*operator norm*, given by

$$\|U\|_\infty \quad := \quad \max_{j=1,\dots,p} \sum_{k=1}^{p} |U_{jk}|, \tag{1}$$

and the $\ell_1/\ell_1$-*operator norm*, given by $\|U\|_1 = \|U^T\|_\infty$. Finally, we use $\|U\|_\infty$ to denote the element-wise maximum $\max_{i,j} |U_{ij}|$; note that this is not a matrix norm, but rather a norm on the vectorized form of the matrix. For any matrix $U \in \mathbb{R}^{p \times p}$, we use $\mathrm{vec}(U)$ or equivalently $\overline{U} \in \mathbb{R}^{p^2}$ to denote its *vectorized form*, obtained by stacking up the rows of $U$. We use $\langle\!\langle U, V \rangle\!\rangle := \sum_{i,j} U_{ij} V_{ij}$ to denote the *trace inner product* on the space of symmetric matrices. Note that this inner product induces the *Frobenius norm* $\|U\|_F := \sqrt{\sum_{i,j} U_{ij}^2}$. Finally, for asymptotics, we use the following standard notation: we write $f(n) = \mathcal{O}(g(n))$ if $f(n) \leq cg(n)$ for some constant $c < \infty$, and $f(n) = \Omega(g(n))$ if $f(n) \geq c'g(n)$ for some constant $c' > 0$. The notation $f(n) \asymp g(n)$ means that $f(n) = \mathcal{O}(g(n))$ and $f(n) = \Omega(g(n))$.

## 2 Background and statement of main result

In this section, we begin by setting up the problem, with some background on Gaussian MRFs and $\ell_1$-regularization. We then state our main result, and discuss some of its consequences.

### 2.1 Gaussian MRFs and $\ell_1$ penalized estimation

Consider an undirected graph $G = (V, E)$ with $p = |V|$ vertices, and let $X = (X_1, \dots, X_p)$ denote a $p$-dimensional Gaussian random vector, with variate $X_i$ identified with vertex $i \in V$. A Gauss-Markov random field (MRF) is described by a density of the form

$$f(x_1, \dots, x_p; \Theta^*) \quad = \quad \frac{1}{(2\pi \det(\Theta^*))^{p/2}} \exp\left\{-\frac{1}{2}x^T \Theta^* x\right\}. \tag{2}$$

As illustrated in Figure 1, Markov structure is reflected in the sparsity pattern of the inverse covariance or concentration matrix $\Theta^*$, a $p \times p$ symmetric matrix. In particular, by the Hammersley-Clifford theorem [7], it must satisfy $\Theta^*_{ij} = 0$ for all $(i, j) \notin E$. Consequently, the problem of graphical model selection is equivalent to estimating the off-diagonal zero-pattern of the concentration matrix—that is, the set $E(\Theta^*) := \{i, j \in V \mid i \neq j, \Theta^*_{ij} \neq 0\}$.

In this paper, we study the minimizer of the $\ell_1$-penalized Gaussian negative log-likelihood. Letting $\langle\!\langle A, B \rangle\!\rangle := \sum_{i,j} A_{ij} B_{ij}$ be the trace inner product on the space of symmetric matrices, this objective function takes the form

$$\widehat{\Theta} = \arg\min_{\Theta \succeq 0} \left\{ \langle\!\langle \Theta, \widehat{\Sigma} \rangle\!\rangle - \mathrm{logdet}(\Theta) + \lambda_n \|\Theta\|_{1,\mathrm{off}} \right\} \quad = \quad \arg\min_{\Theta \succeq 0} g(\Theta; \widehat{\Sigma}, \lambda_n). \tag{3}$$

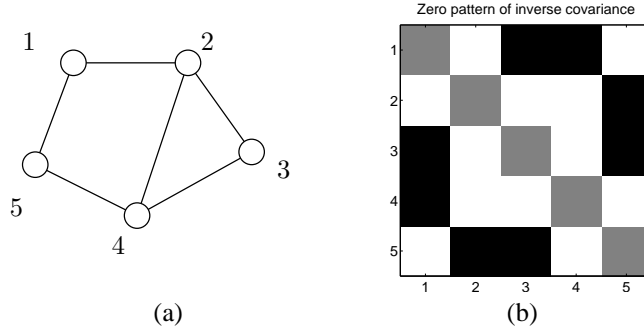

Zero pattern of inverse covariance

(a)                                          (b)

**Figure 1.** (a) Simple undirected graph. A Gauss Markov random field has a Gaussian variable $X_i$ associated with each vertex $i \in V$. This graph has $p = 5$ vertices, maximum degree $d = 3$ and $s = 6$ edges. (b) Zero pattern of the inverse covariance $\Theta^*$ associated with the GMRF in (a). The set $E(\Theta^*)$ corresponds to the off-diagonal non-zeros (white blocks); the diagonal is also non-zero (grey squares), but these entries do not correspond to edges. The black squares correspond to non-edges, or zeros in $\Theta^*$.

Here $\widehat{\Sigma}$ denotes the sample covariance—that is, $\widehat{\Sigma} := \frac{1}{n}\sum_{\ell=1}^{n} X^{(\ell)}[X^{(\ell)}]^T$, where each $X^{(\ell)}$ is drawn in an i.i.d. manner according to the density (2). The quantity $\lambda_n > 0$ is a user-defined regularization parameter. and $\|\Theta\|_{1,\text{off}} := \sum_{i \neq j}|\Theta_{ij}|$ is the *off-diagonal $\ell_1$ regularizer*; note that it does *not* include the diagonal. Since the negative log-determinant is a strictly convex function [5], this problem always has a unique solution, so that there is no ambiguity in equation (3).

We let $E(\widehat{\Theta}) = \{(i,j) \mid i \neq j, \widehat{\Theta}_{ij} \neq 0\}$ denote the edge set associated with the estimate. Of interest in this paper is studying the probability $\mathbb{P}[E(\Theta^*) = E(\widehat{\Theta})]$ as a function of the graph size $p$ (which serves as the "model dimension" for the Gauss-Markov model), the sample size $n$, and the structural properties of $\widehat{\Theta}$. In particular, we define both the *sparsity index*

$$s \quad := \quad |E(\Theta^*)| \quad = \quad \{i,j \in V \mid i \neq j, \Theta_{ij}^* \neq 0\}|. \tag{4}$$

corresponding to the total number of edges, and the *maximum degree or row cardinality*

$$d \quad := \quad \max_{j=1,\ldots,p} |\{i \mid \Theta_{ij}^* \neq 0\}, \tag{5}$$

corresponding to the maximum number of non-zeros in any row of $\Theta^*$, or equivalently the maximum degree in the graph $G$, where we include the diagonal in the degree count.

## 2.2 Statement of main result

Our assumptions involve the Hessian with respect to $\Theta$ of the objective function $g$ defined in equation (3), evaluated at the true model $\Theta^*$. Using standard results on matrix derivatives [5], it can be shown that this Hessian takes the form

$$\Gamma^* \quad := \quad \nabla_\Theta^2 g(\Theta)\Big|_{\Theta=\Theta^*} \quad = \quad \Theta^{*-1} \otimes \Theta^{*-1}, \tag{6}$$

where $\otimes$ denotes the Kronecker matrix product. By definition, $\Gamma^*$ is a $p^2 \times p^2$ matrix indexed by vertex pairs, so that entry $\Gamma^*_{(j,k),(\ell,m)}$ corresponds to the second partial derivative $\frac{\partial^2 g}{\partial \Theta_{jk} \partial \Theta_{\ell m}}$, evaluated at $\Theta = \Theta^*$. When $X$ has multivariate Gaussian distribution, then $\Gamma^*$ is the Fisher information of the model, and by standard results on cumulant functions in exponential families [8], we have the more specific expression $\Gamma^*_{(j,k),(\ell,m)} = \text{cov}\{X_j X_k, X_\ell X_m\}$. For this reason, $\Gamma^*$ can be viewed as an edge-based counterpart to the usual covariance matrix $\Sigma^*$.

We define the set of non-zero off-diagonal entries in the model concentration matrix $\Theta^*$:

$$S(\Theta^*) \quad := \quad \{(i,j) \in V \times V \mid i \neq j, \Theta_{ij}^* \neq 0\}, \tag{7}$$

and let $S(\Theta^*) = \{S(\Theta^*) \cup \{(1,1),\ldots,(p,p)\}$ be the augmented set including the diagonal. We let $S^c(\Theta^*)$ denote the complement of $S(\Theta^*)$ in the set $\{1,\ldots,p\} \times \{1,\ldots,p\}$, corresponding to all

pairs $(\ell, m)$ for which $\Theta_{\ell m}^* = 0$. When it is clear from context, we shorten our notation for these sets to $S$ and $S^c$, respectively. Finally, for any two subsets $T$ and $T'$ of $V \times V$, we use $\Gamma_{TT'}^*$ to denote the $|T| \times |T'|$ matrix with rows and columns of $\Gamma^*$ indexed by $T$ and $T'$ respectively.

We require the following conditions on the Fisher information matrix $\Gamma^*$:

**[A1] Incoherence condition:** This condition captures the intuition that variable-pairs which are non-edges cannot exert an overtly strong effect on variable-pairs which form edges of the Gaussian graphical model.

$$\|\Gamma_{S^cS}^*(\Gamma_{SS}^*)^{-1}\|_\infty \leq (1-\alpha), \quad \text{for some fixed } \alpha > 0. \tag{8}$$

We note that similar conditions arise in the analysis of the Lasso in linear regression [9, 10, 11].

**[A2] Covariance control:** There exist constants $K_{\Sigma^*}, K_{\Gamma^*} < \infty$ such that

$$\|\Theta^{*-1}\|_\infty \leq K_{\Sigma^*}, \quad \text{and} \quad \|(\Gamma_{SS}^*)^{-1}\|_\infty \leq K_{\Gamma^*}. \tag{9}$$

These assumptions require that the covariance elements along any row of $(\Theta^*)^{-1}$ and $(\Gamma_{SS}^*)^{-1}$ have bounded $\ell_1$ norms. Note that similar assumptions are are also required for consistency in Frobenius norm [1].

Recall from equations (4) and (5) the definitions of the sparsity index $s$ and maximum degree $d$, respectively. With this notation, we have:

**Theorem 1.** *Consider a Gaussian distribution with concentration matrix $\Theta^*$ that satisfies conditions (A1) and (A2). Suppose the penalty is set as $\lambda_n = C_1\sqrt{\frac{\log p}{n}}$, and the minimum edge-weight $\Theta_{\min}^* := \min_{(i,j) \in S} |\Theta_{ij}^*|$ scales as $\Theta_{\min}^* > C_2\sqrt{\frac{\log p}{n}}$ for some constants $C_1, C_2 > 0$. Further, suppose the triple $(n, d, p)$ satisfies the scaling*

$$n > L\, d^2 \log(p), \tag{10}$$

*for some constant $L > 0$. Then the edge set $E(\widehat{\Theta})$ specified by the estimator specifies the true edge set w.h.p.—in particular,*

$$\mathbb{P}[E(\widehat{\Theta}) = E(\Theta^*)] \geq 1 - \exp(-c \log p) \to 1. \tag{11}$$

*for some constant $c > 0$.*

**Remarks:** Rothman et al. [1] prove that the error of the estimator in Frobenius norm obeys the bound $\|\widehat{\Theta} - \Theta^*\|_F^2 = \mathcal{O}\{((s+p)\log p)/n\}$, with high probability. We note that model selection consistency does not follow from this result, since an estimate may be close in Frobenius norm while differing substantially in terms of zero-pattern. In one sense, the model selection criterion is more demanding, since given knowledge of the edge set $E(\Theta^*)$, one could restrict estimation procedures to this subset, and so achieve faster rates. On the other hand, Theorem 1 requires incoherence conditions $[A1]$ on the covariance matrix, which are not required for Frobenius norm consistency [1].

## 2.3 Comparison to neighbor-based graphical model selection

It is interesting to compare the estimator to the Gaussian neighborhood regression method studied by Meinshausen and Bühlmann [9], in which each node is linearly regressed with an $\ell_1$ penalty (Lasso) on the rest of the nodes; and the location of the non-zero regression weights is taken as the neighborhood estimate of that node. These neighborhoods are then combined, by either an OR rule or an AND rule, to estimate the full graph. Wainwright [12] shows that the rate $n \asymp d \log p$ is a sharp threshold for the success/failure of neighborhood selection by Lasso. By a union bound over the $p$ nodes, it follows this threshold holds for the Meinshausen and Bühlmann approach as well. This is superior to the scaling in our result (10). However, the two methods rely on slightly different underlying assumptions, and the current form of the neighborhood-based approach requires solving a total of $p$ Lasso programs, as opposed to a single log-determinant problem. Below we show two cases where the Lasso irrepresentability condition holds, while the log-determinant requirement fails. However, in general, we do not know whether the log-determinant irrepresentability strictly dominates its analog for the Lasso.

### 2.3.1 Illustration of irrepresentability: Diamond graph

Consider the following Gaussian MRF example from [13]. Figure 2(a) shows a diamond-shaped graph $G = (V, E)$, with vertex set $V = \{1, 2, 3, 4\}$ and edge-set as the fully connected graph over $V$ with the edge $(1, 4)$ removed. The covariance matrix $\Sigma^*$ is parameterized by the correlation param-

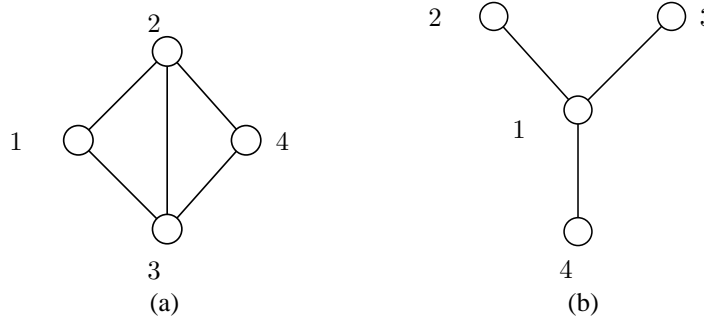

**Figure 2:** (a) Graph of the example discussed by [13]. (b) A simple 4-node star graph.

eter $\rho \in [0, 1/\sqrt{2}]$: the diagonal entries are set to $\Sigma_{ii}^* = 1$, for all $i \in V$; the entries corresponding to edges are set to $\Sigma_{ij}^* = \rho$ for $(i, j) \in E \setminus \{(2, 3)\}$, $\Sigma_{23}^* = 0$; and finally the entry corresponding to the non-edge is set as $\Sigma_{14}^* = 2\rho^2$. For this model, [13] showed that the $\ell_1$-regularized MLE $\widehat{\Theta}$ fails to recover the graph structure for any sample size, if $\rho > -1 + (3/2)^{1/2} \approx 0.23$. It is instructive to compare this necessary condition to the sufficient condition provided in our analysis, namely the incoherence Assumption [$A1$] as applied to the Hessian $\Gamma^*$. For this particular example, a little calculation shows that Assumption [$A1$] is equivalent to the constraint $4|\rho|(|\rho| + 1) < 1$, an inequality which holds for all $\rho \in (-0.2017, 0.2017)$. Note that the upper value $0.2017$ is just below the necessary threshold discussed by [13]. On the other hand, the irrepresentability condition for the Lasso requires only that $2|\rho| < 1$, i.e., $\rho \in (-0.5, 0.5)$. Thus, in the regime $|\rho| \in [0.2017, 0.5)$, the Lasso irrepresentability condition holds while our log-determinant counterpart fails.

### 2.3.2 Illustration of irrepresentability: Star graphs

A second interesting example is the star-shaped graphical model, illustrated in Figure 2(b), which consists of a single hub node connected to the rest of the spoke nodes. We consider a four node graph, with vertex set $V = \{1, 2, 3, 4\}$ and edge-set $E = \{(1, s) \mid s \in \{2, 3, 4\}\}$. The covariance matrix $\Sigma^*$ is parameterized the correlation parameter $\rho \in [-1, 1]$: the diagonal entries are set to $\Sigma_{ii}^* = 1$, for all $i \in V$; the entries corresponding to edges are set to $\Sigma_{ij}^* = \rho$ for $(i, j) \in E$; while the non-edge entries are set as $\Sigma_{ij}^* = \rho^2$ for $(i, j) \notin E$. Consequently, for this particular example, Assumption [$A1$] reduces to the constraint $|\rho|(|\rho| + 2) < 1$, which holds for all $\rho \in (-0.414, 0.414)$. The irrepresentability condition for the Lasso on the other hand allows the full range $\rho \in (-1, 1)$. Thus there is again a regime, $|\rho| \in [0.414, 1)$, where the Lasso irrepresentability condition holds while the log-determinant counterpart fails.

## 3 Proof outline

Theorem 1 follows as a corollary to Theorem 2 in Ravikumar et al [14], an extended and more general version of this paper. There we consider the more general problem of estimation of the covariance matrix of a random vector (that need not necessarily be Gaussian) from i.i.d. samples; and where we relax Assumption [$A2$], and allow quantities $K_{\Sigma^*}$, $K_{\Gamma^*}$ to grow with sample size $n$.

We provide here a high-level outline of the proof of Theorem 1, deferring details to the extended version [14]. Our proofs are based on a technique that we call a *primal-dual witness method*, used previously in analysis of the Lasso [12]. It involves following a specific sequence of steps to construct a pair $(\widetilde{\Theta}, \widetilde{Z})$ of symmetric matrices that together satisfy the optimality conditions associated with the convex program (3) *with high probability*. Thus, when the constructive procedure succeeds, $\widetilde{\Theta}$ is *equal* to the unique solution $\widehat{\Theta}$ of the convex program (3), and $\widetilde{Z}$ is an optimal solution to its

dual. In this way, the estimator $\widehat{\Theta}$ inherits from $\widetilde{\Theta}$ various optimality properties in terms of its distance to the truth $\Theta^*$, and its recovery of the signed sparsity pattern. To be clear, our procedure for constructing $\widetilde{\Theta}$ is *not* a practical algorithm for solving the log-determinant problem (3), but rather is used as a proof technique for certifying the behavior of the $\ell_1$-regularized MLE (3).

## 3.1 Primal-dual witness approach

At the core of the primal-dual witness method are the standard convex optimality conditions that characterize the optimum $\widehat{\Theta}$ of the convex program (3). For future reference, we note that the sub-differential of the norm $\|\cdot\|_{1,\mathrm{off}}$ evaluated at some $\Theta$ consists the set of all symmetric matrices $Z \in \mathbb{R}^{p \times p}$ such that

$$
Z_{ij} = \begin{cases} 0 & \text{if } i = j \\ \mathrm{sign}(\Theta_{ij}) & \text{if } i \neq j \text{ and } \Theta_{ij} \neq 0 \\ \in [-1, +1] & \text{if } i \neq j \text{ and } \Theta_{ij} = 0. \end{cases} \tag{12}
$$

**Lemma 1.** *For any $\lambda_n > 0$ and sample covariance $\widehat{\Sigma}$ with strictly positive diagonal, the $\ell_1$-regularized log-determinant problem (3) has a unique solution $\widehat{\Theta} \succ 0$ characterized by*

$$
\widehat{\Sigma} - \widehat{\Theta}^{-1} + \lambda_n \widetilde{Z} = 0, \tag{13}
$$

*where $\widetilde{Z}$ is an element of the subdifferential $\partial\|\widehat{\Theta}\|_{1,\mathrm{off}}$.*

Based on this lemma, we construct the primal-dual witness solution $(\widetilde{\Theta}, \widetilde{Z})$ as follows:

(a) We determine the matrix $\widetilde{\Theta}$ by solving the restricted log-determinant problem

$$
\widetilde{\Theta} := \arg \min_{\Theta \succ 0, \, \Theta_{S^c} = 0} \left\{ \langle\!\langle \Theta, \, \widehat{\Sigma} \rangle\!\rangle - \log \det(\Theta) + \lambda_n \|\Theta\|_{1,\mathrm{off}} \right\}. \tag{14}
$$

Note that by construction, we have $\widetilde{\Theta} \succ 0$, and moreover $\widetilde{\Theta}_{S^c} = 0$.

(b) We choose $\widetilde{Z}_S$ as a member of the sub-differential of the regularizer $\|\cdot\|_{1,\mathrm{off}}$, evaluated at $\widetilde{\Theta}$.

(c) We set $\widetilde{Z}_{S^c}$ as

$$
\widetilde{Z}_{S^c} = \frac{1}{\lambda_n} \left\{ -\widehat{\Sigma}_{S^c} + [\widetilde{\Theta}^{-1}]_{S^c} \right\}, \tag{15}
$$

which ensures that constructed matrices $(\widetilde{\Theta}, \widetilde{Z})$ satisfy the optimality condition (13).

(d) We verify the *strict dual feasibility* condition

$$
|\widetilde{Z}_{ij}| < 1 \quad \text{for all } (i, j) \in S^c.
$$

To clarify the nature of the construction, steps (a) through (c) suffice to obtain a pair $(\widetilde{\Theta}, \widetilde{Z})$ that satisfy the optimality conditions (13), but do *not* guarantee that $\widetilde{Z}$ is an element of sub-differential $\partial\|\widetilde{\Theta}\|_{1,\mathrm{off}}$. By construction, specifically step (b) of the construction ensures that the entries $\widetilde{Z}$ in $S$ satisfy the sub-differential conditions, since $\widetilde{Z}_S$ is a member of the sub-differential of $\partial\|\widetilde{\Theta}_S\|_{1,\mathrm{off}}$. The purpose of step (d), then, is to verify that the remaining elements of $\widetilde{Z}$ satisfy the necessary conditions to belong to the sub-differential.

If the primal-dual witness construction succeeds, then it acts as a *witness* to the fact that the solution $\widetilde{\Theta}$ to the restricted problem (14) is equivalent to the solution $\widehat{\Theta}$ to the original (unrestricted) problem (3). We exploit this fact in our proof of Theorem 1: we first show that the primal-dual witness technique succeeds with high-probability, from which we can conclude that the support of the optimal solution $\widehat{\Theta}$ is contained within the support of the true $\Theta^*$. The next step requires checking that none of the entries in $\widetilde{\Theta}_S$ constructed in Equation (14) are zero. It is to verify this that we require the lower bound assumption in Theorem 1 on the value of the minimum value $\Theta^*_{\min}$.

# 4 Experiments

In this section, we describe some experiments which illustrate the model selection rates in Theorem 1. We solved the $\ell_1$ penalized log-determinant optimization problem using the "glasso" program [4], which builds on the block co-ordinate descent algorithm of [3]. We report experiments for star-shaped graphs, which consist of one node connected to the rest of the nodes. These graphs allow us to vary both $d$ and $p$, since the degree of the central hub can be varied between 1 and $p-1$. Applying the algorithm to these graphs should therefore provide some insight on how the required number of samples $n$ is related to $d$ and $p$. We tested varying graph sizes $p$ from $p = 64$ upwards to $p = 375$. The edge-weights were set as entries in the inverse of a covariance matrix $\Sigma^*$ with diagonal entries set as $\Sigma_{ii}^* = 1$ for all $i = 1, \ldots, p$, and $\Sigma_{ij}^* = 2.5/d$ for all $(i,j) \in E$, so that the quantities $(K_{\Sigma^*}, K_{\Gamma^*}, \alpha)$ remain constant.

**Dependence on graph size:**

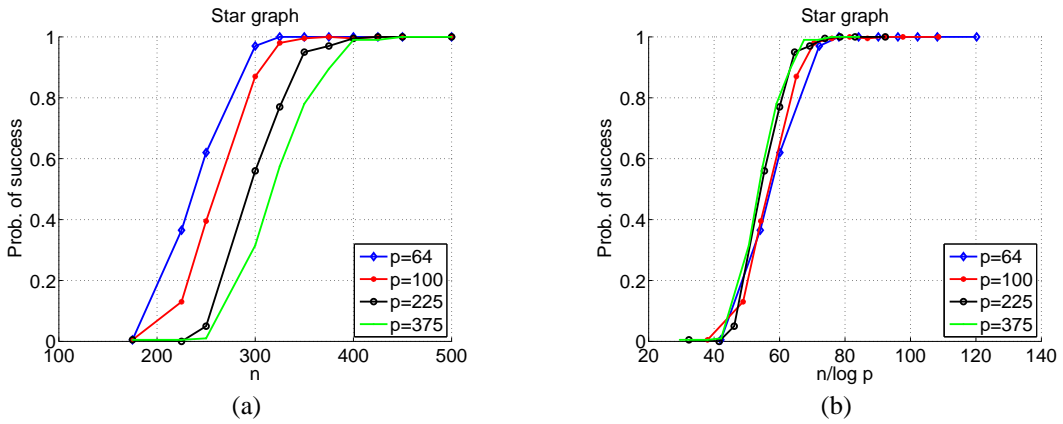

**Figure 3.** Simulations for a star graph with varying number of nodes $p$, fixed maximal degree $d = 40$, and edge covariances $\Sigma_{ij}^* = 1/16$ for all edges. Plots of probability of correct signed edge-set recovery versus the sample size $n$ in panel (a), and versus the rescaled sample size $n/\log p$ in panel (b). Each point corresponds to the average over $N = 100$ trials.

Panel (a) of Figure 3 plots the probability of correct signed edge-set recovery against the sample size $n$ for a star-shaped graph of three different graph sizes $p$. For each curve, the probability of success starts at zero (for small sample sizes $n$), but then transitions to one as the sample size is increased. As would be expected, it is more difficult to perform model selection for larger graph sizes, so that (for instance) the curve for $p = 375$ is shifted to the right relative to the curve for $p = 64$. Panel (b) of Figure 3 replots the same data, with the horizontal axis rescaled by $(1/\log p)$. This scaling was chosen because our theory predicts that the sample size should scale logarithmically with $p$ (see equation (10)). Consistent with this prediction, when plotted against the rescaled sample size $n/\log p$, the curves in panel (b) all stack up. Consequently, the ratio $(n/\log p)$ acts as an effective sample size in controlling the success of model selection, consistent with the predictions of Theorem 1.

**Dependence on the maximum node degree:**

Panel (a) of Figure 4 plots the probability of correct signed edge-set recovery against the sample size $n$ for star-shaped graphs; each curve corresponds to a different choice of maximum node degree $d$, allowing us to investigate the dependence of the sample size on this parameter. So as to control these comparisons, we fixed the number of nodes to $p = 200$. Observe how the plots in panel (a) shift to the right as the maximum node degree $d$ is increased, showing that star-shaped graphs with higher degrees are more difficult. In panel (b) of Figure 4, we plot the same data versus the rescaled sample size $n/d$. Recall that if all the curves were to stack up under this rescaling, then it means the required sample size $n$ scales linearly with $d$. These plots are closer to aligning than the unrescaled plots, but the agreement is not perfect. In particular, observe that the curve $d$ (right-most in panel (a)) remains a bit to the right in panel (b), which suggests that a somewhat more aggressive rescaling—perhaps $n/d^\gamma$ for some $\gamma \in (1, 2)$—is appropriate. The sufficient condition from Theorem 1, as summarized

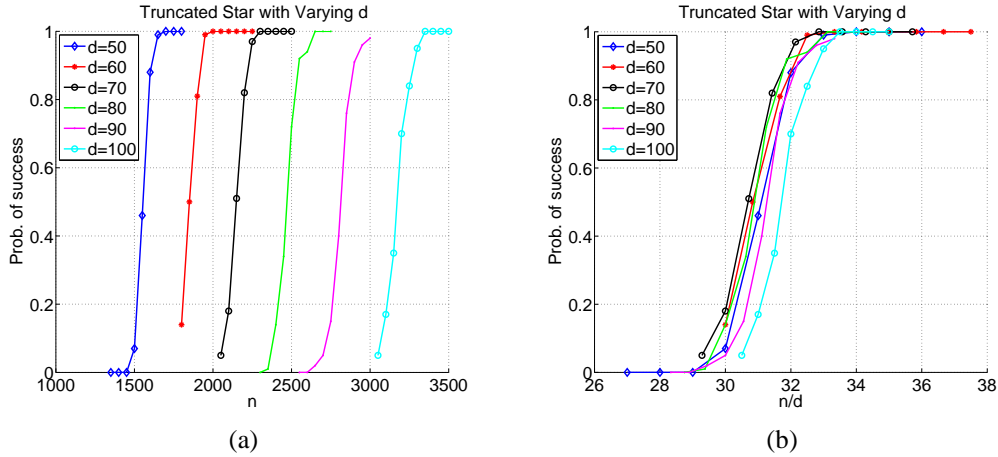

**Figure 4.** Simulations for star graphs with fixed number of nodes $p = 200$, varying maximal (hub) degree $d$, edge covariances $\Sigma_{ij}^* = 2.5/d$. Plots of probability of correct signed edge-set recovery versus the sample size $n$ in panel (a), and versus the rescaled sample size $n/d$ in panel (b).

in equation (10), is $n = \Omega(d^2 \log p)$, which appears to be overly conservative based on these data. Thus, it might be possible to tighten our theory under certain regimes.

## References

[1] A.J. Rothman, P.J. Bickel, E. Levina, and J. Zhu. Sparse permutation invariant covariance estimation. *Electron. J. Statist.*, 2:494–515, 2008.

[2] M. Yuan and Y. Lin. Model selection and estimation in the Gaussian graphical model. *Biometrika*, 94(1):19–35, 2007.

[3] A. d'Asprémont, O. Banerjee, and L. El Ghaoui. First-order methods for sparse covariance selection. *SIAM J. Matrix Anal. Appl.*, 30(1):56–66, 2008.

[4] J. Friedman, T. Hastie, and R. Tibshirani. Sparse inverse covariance estimation with the graphical Lasso. *Biostat.*, 9(3):432–441, 2007.

[5] S. Boyd and L. Vandenberghe. *Convex optimization*. Cambridge University Press, Cambridge, UK, 2004.

[6] R. A. Horn and C. R. Johnson. *Matrix Analysis*. Cambridge University Press, Cambridge, 1985.

[7] S. L. Lauritzen. *Graphical Models*. Oxford University Press, Oxford, 1996.

[8] L.D. Brown. *Fundamentals of statistical exponential families*. Institute of Mathematical Statistics, Hayward, CA, 1986.

[9] N. Meinshausen and P. Bühlmann. High-dimensional graphs and variable selection with the Lasso. *Ann. Statist.*, 34(3):1436–1462, 2006.

[10] J. A. Tropp. Just relax: Convex programming methods for identifying sparse signals. *IEEE Trans. Info. Theory*, 51(3):1030–1051, 2006.

[11] P. Zhao and B. Yu. On model selection consistency of Lasso. *Journal of Machine Learning Research*, 7:2541–2567, 2006.

[12] M. J. Wainwright. Sharp thresholds for high-dimensional and noisy recovery of sparsity using the Lasso. Technical Report 709, UC Berkeley, May 2006. To appear in IEEE Trans. Info. Theory.

[13] N. Meinshausen. A note on the Lasso for graphical Gaussian model selection. *Statistics and Probability Letters*, 78(7):880–884, 2008.

[14] P. Ravikumar, M. J. Wainwright, G. Raskutti, and B. Yu. High-dimensional covariance estimation by minimizing $\ell_1$-penalized log-determinant divergence. Technical Report 767, Department of Statistics, UC Berkeley, November 2008.

